# Algorithmic Luckiness

**Ralf Herbrich**
Microsoft Research Ltd.
CB3 0FB Cambridge
United Kingdom
*rherb@microsoft.com*

**Robert C. Williamson**
Australian National University
Canberra 0200
Australia
*Bob.Williamson@anu.edu.au*

## Abstract

In contrast to standard statistical learning theory which studies uniform bounds on the expected error we present a framework that exploits the specific learning algorithm used. Motivated by the luckiness framework [8] we are also able to exploit the serendipity of the training sample. The main difference to previous approaches lies in the complexity measure; rather than covering all hypotheses in a given hypothesis space it is only necessary to cover the functions which could have been learned using the fixed learning algorithm. We show how the resulting framework relates to the VC, luckiness and compression frameworks. Finally, we present an application of this framework to the maximum margin algorithm for linear classifiers which results in a bound that exploits both the margin and the distribution of the data in feature space.

## 1  Introduction

Statistical learning theory is mainly concerned with the study of *uniform* bounds on the expected error of hypotheses from a given hypothesis space [9, 1]. Such bounds have the appealing feature that they provide performance guarantees for classifiers found by *any* learning algorithm. However, it has been observed that these bounds tend to be overly pessimistic. One explanation is that only in the case of learning algorithms which minimise the training error it has been proven that uniformity of the bounds is equivalent to studying the learning algorithm's generalisation performance directly.

In this paper we present a theoretical framework which aims at *directly* studying the generalisation error of a learning algorithm rather than taking the detour via the uniform convergence of training errors to expected errors in a given hypothesis space. In addition, our new model of learning allows the exploitation of the fact that we serendipitously observe a training sample which is easy to learn by a given learning algorithm. In that sense, our framework is a descendant of the luckiness framework of Shawe-Taylor et al. [8]. In the present case, the luckiness is a function of a given learning algorithm and a given training sample and characterises the diversity of the algorithms solutions. The notion of luckiness allows us to study given learning algorithms at many different perspectives. For example, the maximum margin algorithm [9] can either been studied via the number of dimensions in feature space,

the margin of the classifier learned or the sparsity of the resulting classifier. Our main results are two generalisation error bounds for learning algorithms: one for the zero training error scenario and one agnostic bound (Section 2). We shall demonstrate the usefulness of our new framework by studying its relation to the VC framework, the original luckiness framework and the compression framework of Littlestone and Warmuth [6] (Section 3). Finally, we present an application of the new framework to the maximum margin algorithm for linear classifiers (Section 4). The detailed proofs of our main results can be found in [5].

We denote vectors using bold face, e.g. $\boldsymbol{x} = (x_1, \ldots, x_m)$ and the length of this vector by $|\boldsymbol{x}|$, i.e. $|\boldsymbol{x}| = m$. In order to unburden notation we use the shorthand notation $\boldsymbol{z}_{[i:j]} := (z_i, \ldots, z_j)$ for $i \leq j$. Random variables are typeset in sans-serif font. The symbols $\mathbf{P}_\mathsf{X}$, $\mathbf{E}_\mathsf{X}[f(\mathsf{X})]$ and $\mathbb{I}$ denote a probability measure over $\mathsf{X}$, the expectation of $f(\cdot)$ over the random draw of its argument $x$ and the indicator function, respectively. The shorthand notation $\mathcal{Z}^{(\infty)} := \cup_{m=1}^{\infty} \mathcal{Z}^m$ denotes the union of all $m$–fold Cartesian products of the set $\mathcal{Z}$. For any $m \in \mathbb{N}$ we define $I_m \subset \{1, \ldots, m\}^m$ as the set of all permutations of the numbers $1, \ldots, m$,

$$I_m := \{(i_1, \ldots, i_m) \in \{1, \ldots, m\}^m \mid \forall j \neq k : i_j \neq i_k\} .$$

Given a $2m$–vector $\boldsymbol{i} \in I_{2m}$ and a sample $\boldsymbol{z} \in \mathcal{Z}^{2m}$ we define $\pi_{\boldsymbol{i}} : \{1, \ldots, 2m\} \to \{1, \ldots, 2m\}$ by $\pi_{\boldsymbol{i}}(j) := i_j$ and $\Pi_{\boldsymbol{i}}(\boldsymbol{z})$ by $\Pi_{\boldsymbol{i}}(\boldsymbol{z}) := \left(z_{\pi_{\boldsymbol{i}}(1)}, \ldots, z_{\pi_{\boldsymbol{i}}(2m)}\right)$.

## 2 Algorithmic Luckiness

Suppose we are given a training sample $\boldsymbol{z} = (\boldsymbol{x}, \boldsymbol{y}) \in (\mathcal{X} \times \mathcal{Y})^m = \mathcal{Z}^m$ of size $m \in \mathbb{N}$ independently drawn (iid) from some unknown but fixed distribution $\mathbf{P}_\mathsf{XY} = \mathbf{P}_\mathsf{Z}$ together with a learning algorithm $\mathcal{A} : \mathcal{Z}^{(\infty)} \to \mathcal{Y}^{\mathcal{X}}$. For a predefined loss $l : \mathcal{Y} \times \mathcal{Y} \to [0,1]$ we would like to investigate the generalisation error $G_l[\mathcal{A}, \boldsymbol{z}] := R_l[\mathcal{A}(\boldsymbol{z})] - \inf_{h \in \mathcal{Y}^\mathcal{X}} R_l[h]$ of the algorithm where the *expected error* $R_l[h]$ of $h$ is defined by

$$R_l[h] := \mathbf{E}_\mathsf{XY}[l(h(\mathsf{X}), \mathsf{Y})] .$$

Since $\inf_{h \in \mathcal{Y}^\mathcal{X}} R_l[h]$ (which is also known as the *Bayes error*) is independent of $\mathcal{A}$ it suffices to bound $R_l[\mathcal{A}(\boldsymbol{z})]$. Although we know that for any fixed hypothesis $h$ the *training error*

$$\widehat{R}_l[h, \boldsymbol{z}] := \frac{1}{|\boldsymbol{z}|} \sum_{(x_i, y_i) \in \boldsymbol{z}} l(h(x_i), y_i)$$

is with high probability (over the random draw of the training sample $\boldsymbol{z} \in \mathcal{Z}^{(\infty)}$) close to $R_l[h]$, this might no longer be true for the *random* hypothesis $\mathcal{A}(\boldsymbol{z})$. Hence we would like to state that with only small probability (at most $\delta$), the expected error $R_l[\mathcal{A}(\boldsymbol{z})]$ is larger than the training error $\widehat{R}_l[\mathcal{A}(\boldsymbol{z}), \boldsymbol{z}]$ plus some sample *and* algorithm dependent complexity $\varepsilon(\mathcal{A}, \boldsymbol{z}, \delta)$,

$$\mathbf{P}_{\mathsf{Z}^m}\left(R_l[\mathcal{A}(\mathsf{Z})] > \widehat{R}_l[\mathcal{A}(\mathsf{Z}), \mathsf{Z}] + \varepsilon(\mathcal{A}, \mathsf{Z}, \delta)\right) < \delta . \tag{1}$$

In order to derive such a bound we utilise a modified version of the basic lemma of Vapnik and Chervonenkis [10].

**Lemma 1.** *For all loss functions $l : \mathcal{Y} \times \mathcal{Y} \to [0,1]$, all probability measures $\mathbf{P}_\mathsf{Z}$, all algorithms $\mathcal{A}$ and all measurable formulas $\Upsilon : \mathcal{Z}^m \to \{\text{true}, \text{false}\}$, if $m\varepsilon^2 > 2$ then*

$$\mathbf{P}_{\mathsf{Z}^m}\left(\left(R_l[\mathcal{A}(\mathsf{Z})] > \widehat{R}_l[\mathcal{A}(\mathsf{Z}), \mathsf{Z}] + \varepsilon\right) \wedge \Upsilon(\mathsf{Z})\right) <$$

$$2\underbrace{\mathbf{P}_{\mathsf{Z}^{2m}}\left(\left(\widehat{R}_l[\mathcal{A}(\mathsf{Z}_{[1:m]}), \mathsf{Z}_{[(m+1):2m]}] > \widehat{R}_l[\mathcal{A}(\mathsf{Z}_{[1:m]}), \mathsf{Z}_{[1:m]}] + \frac{\varepsilon}{2}\right) \wedge \Upsilon(\mathsf{Z}_{[1:m]})\right)}_{J(\mathsf{Z})} .$$

*Proof (Sketch).* The probability on the r.h.s. is lower bounded by the probability of the conjunction of event on the l.h.s. and $Q(z) \equiv R_l \left[ \mathcal{A} \left( z_{[1:m]} \right) \right] - \widehat{R}_l \left[ \mathcal{A} \left( z_{[1:m]} \right), z_{(m+1):2m} \right] < \frac{\varepsilon}{2}$. Note that this probability is over $z \in \mathcal{Z}^{2m}$. If we now condition on the first $m$ examples, $\mathcal{A} \left( z_{[1:m]} \right)$ is fixed and therefore by an application of Hoeffding's inequality (see, e.g. [1]) and since $m\varepsilon^2 > 2$ the additional event $Q$ has probability of at least $\frac{1}{2}$ over the random draw of $(z_{m+1}, \ldots, z_{2m})$. □

Use of Lemma 1 — which is similar to the approach of classical VC analysis — reduces the original problem (1) to the problem of studying the deviation of the training errors on the first and second half of a double sample $z \in \mathcal{Z}^{2m}$ of size $2m$. It is of utmost importance that the hypothesis $\mathcal{A} \left( z_{[1:m]} \right)$ is always learned from the first $m$ examples. Now, in order to fully exploit our assumptions of the mutual independence of the double sample $z \in \mathcal{Z}^{2m}$ we use a technique known as symmetrisation by permutation: since $\mathbf{P}_{Z^{2m}}$ is a product measure, it has the property that $\mathbf{P}_{Z^{2m}} (J(\mathbf{Z})) = \mathbf{P}_{Z^{2m}} (J(\Pi_i(\mathbf{Z})))$ for any $i \in I_{2m}$. Hence, it suffices to bound the probability of permutations $\pi_i$ such that $J(\Pi_i(z))$ is true for a given and *fixed* double sample $z$. As a consequence thereof, we only need to count the number of different hypotheses that can be learned by $\mathcal{A}$ from the first $m$ examples when permuting the double sample.

**Definition 1 (Algorithmic luckiness).** Any function $L$ that maps an algorithm $\mathcal{A} : \mathcal{Z}^{(\infty)} \to \mathcal{Y}^{\mathcal{X}}$ and a training sample $z \in \mathcal{Z}^{(\infty)}$ to a real value is called an *algorithmic luckiness*. For all $m \in \mathbb{N}$, for any $z \in \mathcal{Z}^{2m}$, the *lucky set* $\mathcal{H}_\mathcal{A}(L, z) \subseteq \mathcal{Y}^{\mathcal{X}}$ is the set of all hypotheses that are learned from the first $m$ examples $\left( z_{\pi_i(1)}, \ldots, z_{\pi_i(m)} \right)$ when permuting the whole sample $z$ whilst not decreasing the luckiness, i.e.

$$\mathcal{H}_\mathcal{A}(L, z) := \left\{ \mathcal{A} \left( z_{\pi_i(1)}, \ldots, z_{\pi_i(m)} \right) \mid i \in \mathcal{I}_\mathcal{A}(L, z) \right\}, \tag{2}$$

where

$$\mathcal{I}_\mathcal{A}(L, z) := \left\{ i \in I_{2m} \mid L \left( \mathcal{A}, \left( z_{\pi_i(1)}, \ldots, z_{\pi_i(m)} \right) \right) \geq L(\mathcal{A}, (z_1, \ldots, z_m)) \right\}. \tag{3}$$

Given a fixed loss function $l : \mathcal{Y} \times \mathcal{Y} \to [0,1]$ the *induced loss function set* $\mathcal{L}_l (\mathcal{H}_\mathcal{A}(L, z))$ is defined by

$$\mathcal{L}_l (\mathcal{H}_\mathcal{A}(L, z)) := \left\{ (x, y) \mapsto l(h(x), y) \mid h \in \mathcal{H}_\mathcal{A}(L, z) \right\}.$$

For any luckiness function $L$ and any learning algorithm $\mathcal{A}$, the complexity of the double sample $z$ is the minimal number $\mathcal{N}_1 (\tau, \mathcal{L}_l (\mathcal{H}_\mathcal{A}(L, z)), z)$ of hypotheses $\hat{h} \in \mathcal{Y}^{\mathcal{X}}$ needed to cover $\mathcal{L}_l (\mathcal{H}_\mathcal{A}(L, z))$ at some predefined scale $\tau$, i.e. for any hypothesis $h \in \mathcal{H}_\mathcal{A}(L, z)$ there exists a $\hat{h} \in \mathcal{Y}^{\mathcal{X}}$ such that

$$\frac{1}{2m} \sum_{i=1}^{2m} \left| l(h(x_i), y_i) - l \left( \hat{h}(x_i), y_i \right) \right| \leq \tau. \tag{4}$$

To see this note that whenever $J(\Pi_i(z))$ is true (over the random draw of permutations) then there exists a function $\hat{h}$ which has a difference in the training errors on the double sample of at least $\frac{\varepsilon}{2} + 2\tau$. By an application of the union bound we see that the number $\mathcal{N}_1 (\tau, \mathcal{L}_l (\mathcal{H}_\mathcal{A}(L, z)), z)$ is of central importance. Hence, if we are able to bound this number over the random draw of the double sample $z$ only using the luckiness on the first $m$ examples we can use this bound in place of the worst case complexity $\sup_{z \in \mathcal{Z}^{2m}} \mathcal{N}_1 (\tau, \mathcal{L}_l (\mathcal{H}_\mathcal{A}(L, z)), z)$ as usually done in the VC framework (see [9]).

**Definition 2 ($\omega$–smallness of $L$).** Given an algorithm $\mathcal{A} : \mathcal{Z}^{(\infty)} \to \mathcal{Y}^{\mathcal{X}}$ and a loss $l : \mathcal{Y} \times \mathcal{Y} \to [0,1]$ the algorithmic luckiness function $L$ is $\omega$–*small at scale* $\tau \in \mathbb{R}^+$ if for all $m \in \mathbb{N}$, all $\delta \in (0,1]$ and all $\mathbf{P}_Z$

$$\mathbf{P}_{Z^{2m}} \underbrace{\left( \mathcal{N}_1 \left( \tau, \mathcal{L}_l \left( \mathcal{H}_{\mathcal{A}} \left( L, \mathbf{Z} \right) \right), \mathbf{Z} \right) > \omega \left( L \left( \mathcal{A}, \mathbf{Z}_{[1:m]} \right), l, m, \delta, \tau \right) \right)}_{S(\mathbf{Z})} < \delta \,.$$

Note that if the range of $l$ is $\{0,1\}$ then $\mathcal{N}_1 \left( \frac{1}{2m}, \mathcal{L}_l \left( \mathcal{H}_{\mathcal{A}} \left( L, \boldsymbol{z} \right) \right), \boldsymbol{z} \right)$ equals the number of dichotomies on $\boldsymbol{z}$ incurred by $\mathcal{L}_l \left( \mathcal{H}_{\mathcal{A}} \left( L, \boldsymbol{z} \right) \right)$.

---

**Theorem 1 (Algorithmic luckiness bounds).** *Suppose we have a learning algorithm $\mathcal{A} : \mathcal{Z}^{(\infty)} \to \mathcal{Y}^{\mathcal{X}}$ and an algorithmic luckiness $L$ that is $\omega$–small at scale $\tau$ for a loss function $l : \mathcal{Y} \times \mathcal{Y} \to [0,1]$. For any probability measure $\mathbf{P}_Z$, any $d \in \mathbb{N}$ and any $\delta \in (0,1]$, with probability at least $1 - \delta$ over the random draw of the training sample $\boldsymbol{z} \in \mathcal{Z}^m$ of size $m$, if $\omega \left( L \left( \mathcal{A}, \boldsymbol{z} \right), l, m, \delta/4, \tau \right) \leq 2^d$ then*

$$R_l \left[ \mathcal{A} \left( \boldsymbol{z} \right) \right] \leq \widehat{R}_l \left[ \mathcal{A} \left( \boldsymbol{z} \right), \boldsymbol{z} \right] + \sqrt{\frac{8}{m} \left( d + \log_2 \left( \frac{4}{\delta} \right) \right)} + 4\tau \,. \tag{5}$$

*Furthermore, under the above conditions if the algorithmic luckiness $L$ is $\omega$–small at scale $\frac{1}{2m}$ for a binary loss function $l \left( \cdot, \cdot \right) \in \{0,1\}$ and $\widehat{R}_l \left[ \mathcal{A} \left( \boldsymbol{z} \right), \boldsymbol{z} \right] = 0$ then*

$$R_l \left[ \mathcal{A} \left( \boldsymbol{z} \right) \right] \leq \frac{2}{m} \left( d + \log_2 \left( \frac{4}{\delta} \right) \right) \,. \tag{6}$$

---

*Proof (Compressed Sketch).* We will only sketch the proof of equation (5); the proof of (6) is similar and can be found in [5]. First, we apply Lemma 1 with $\Upsilon \left( \boldsymbol{z} \right) \equiv \omega \left( L \left( \mathcal{A}, \boldsymbol{z} \right), l, m, \delta/4, \tau \right) \leq 2^d$. We now exploit the fact that

$$\mathbf{P}_{Z^{2m}} \left( J \left( \mathbf{Z} \right) \right) = \underbrace{\mathbf{P}_{Z^{2m}} \left( J \left( \mathbf{Z} \right) \wedge S \left( \mathbf{Z} \right) \right)}_{\leq \mathbf{P}_{Z^{2m}} \left( S(\mathbf{Z}) \right)} + \mathbf{P}_{Z^{2m}} \left( J \left( \mathbf{Z} \right) \wedge \neg S \left( \mathbf{Z} \right) \right)$$

$$< \frac{\delta}{4} + \mathbf{P}_{Z^{2m}} \left( J \left( \mathbf{Z} \right) \wedge \neg S \left( \mathbf{Z} \right) \right),$$

which follows from Definition 2. Following the above-mentioned argument it suffices to bound the probability of a random permutation $\Pi_{\mathsf{I}} \left( \boldsymbol{z} \right)$ that $J \left( \Pi_{\mathsf{I}} \left( \boldsymbol{z} \right) \right) \wedge \neg S \left( \Pi_{\mathsf{I}} \left( \boldsymbol{z} \right) \right)$ is true for a *fixed* double sample $\boldsymbol{z}$. Noticing that $\Upsilon \left( \boldsymbol{z} \right) \wedge \neg S \left( \boldsymbol{z} \right) \Rightarrow \mathcal{N}_1 \left( \tau, \mathcal{L}_l \left( \mathcal{H}_{\mathcal{A}} \left( L, \boldsymbol{z} \right) \right), \boldsymbol{z} \right) \leq 2^d$ we see that we only consider swappings $\pi_i$ for which $\mathcal{N}_1 \left( \tau, \mathcal{L}_l \left( \mathcal{H}_{\mathcal{A}} \left( L, \Pi_i \left( \boldsymbol{z} \right) \right) \right), \Pi_i \left( \boldsymbol{z} \right) \right) \leq 2^d$. Thus let us consider such a cover of size not more than $2^d$. By (4) we know that whenever $J \left( \Pi_i \left( \boldsymbol{z} \right) \right) \wedge \neg S \left( \Pi_i \left( \boldsymbol{z} \right) \right)$ is true for a swapping $i$ then there exists a hypothesis $\hat{h} \in \mathcal{Y}^{\mathcal{X}}$ in the cover such that $\widehat{R}_l \left[ \hat{h}, \left( \Pi_{\mathsf{I}} \left( \boldsymbol{z} \right) \right)_{[(m+1):2m]} \right] - \widehat{R}_l \left[ \hat{h}, \left( \Pi_{\mathsf{I}} \left( \boldsymbol{z} \right) \right)_{[1:m]} \right] > \frac{\varepsilon}{2} + 2\tau$. Using the union bound and Hoeffding's inequality for a particular choice of $\mathbf{P}_{\mathsf{I}}$ shows that $\mathbf{P}_{\mathsf{I}} \left( J \left( \Pi_{\mathsf{I}} \left( \boldsymbol{z} \right) \right) \wedge \neg S \left( \Pi_{\mathsf{I}} \left( \boldsymbol{z} \right) \right) \right) \leq \frac{\delta}{4}$ which finalises the proof. $\qquad\square$

A closer look at (5) and (6) reveals that the essential difference to uniform bounds on the expected error is within the definition of the covering number: rather than covering all hypotheses $h$ in a given hypothesis space $\mathcal{H} \subseteq \mathcal{Y}^{\mathcal{X}}$ for a given double sample it suffices to cover all hypotheses that can be learned by a given learning algorithm from the first half when permuting the double sample. Note that the usage of permutations in the definition of (2) is not only a technical matter; it fully exploits all the assumptions made for the training sample, namely the training sample is drawn iid.

## 3  Relationship to Other Learning Frameworks

In this section we present the relationship of algorithmic luckiness to other learning frameworks (see [9, 8, 6] for further details of these frameworks).

**VC Framework**  If we consider a binary loss function $l\left(\cdot,\cdot\right)\in\{0,1\}$ and assume that the algorithm $\mathcal{A}$ selects functions from a given hypothesis space $\mathcal{H}\subseteq\mathcal{Y}^{\mathcal{X}}$ then $L\left(\mathcal{A},z\right)=-\mathrm{VCDim}\left(\mathcal{H}\right)$ is a $\omega$–small luckiness function where

$$\omega\left(L_0,l,m,\delta,\frac{1}{2m}\right)\leq\left(\frac{2em}{-L_0}\right)^{-L_0}. \tag{7}$$

This can easily be seen by noticing that the latter term is an upper bound on $\max_{z\in\mathcal{Z}^{2m}}\left|\{\left(l\left(h\left(x_1\right),y_1\right),\ldots,l\left(h\left(x_{2m}\right),y_{2m}\right)\right):h\in\mathcal{H}\}\right|$ (see also [9]). Note that this luckiness function neither exploits the particular training sample observed nor the learning algorithm used.

**Luckiness Framework**  Firstly, the luckiness framework of Shawe-Taylor et al. [8] only considered binary loss functions $l$ and the zero training error case. In this work, the luckiness $\tilde{L}$ is a function of hypothesis and training samples and is called $\tilde{\omega}$–small if the probability over the random draw of a $2m$ sample $z$ that there exists a hypothesis $h$ with $\tilde{\omega}(\tilde{L}(h,(z_1,\ldots,z_m)),\delta)<\mathcal{N}_1(\frac{1}{2m},\{(x,y)\mapsto l\left(g\left(x\right),y\right)\mid\tilde{L}\left(g,z\right)\geq\tilde{L}\left(h,z\right)\},z)$, is smaller than $\delta$. Although similar in spirit, the classical luckiness framework does not allow exploitation of the learning algorithm used to the same extent as our new luckiness. In fact, in this framework not only the covering number must be estimable but also the variation of the luckiness $\tilde{L}$ itself. These differences make it very difficult to formally relate the two frameworks.

**Compression Framework**  In the compression framework of Littlestone and Warmuth [6] one considers learning algorithms $\mathcal{A}$ which are compression schemes, i.e. $\mathcal{A}\left(z\right)=\mathcal{R}\left(\mathcal{C}\left(z\right)\right)$ where $\mathcal{C}\left(z\right)$ selects a subsample $\bar{z}\subseteq z$ and $\mathcal{R}:\mathcal{Z}^{(\infty)}\to\mathcal{Y}^{\mathcal{X}}$ is a permutation invariant reconstruction function. For this class of learning algorithms, the luckiness $L\left(\mathcal{A},z\right)=-\left|\mathcal{C}\left(z\right)\right|$ is $\omega$–small where $\omega$ is given by (7). In order to see this we note that (3) ensures that we only consider permutations $\pi_i$ where $\mathcal{C}\left(\Pi_i\left(z\right)\right)\leq\left|\mathcal{C}\left(z\right)\right|$, i.e. we use not more than $-L$ training examples from $z\in\mathcal{Z}^{2m}$. As there are exactly $\binom{2m}{d}$ distinct choices of $d$ training examples from $2m$ examples the result follows by application of Sauer's lemma [9]. Disregarding constants, Theorem 1 gives exactly the same bound as in [6].

## 4  A New Margin Bound For Support Vector Machines

In this section we study the maximum margin algorithm for linear classifiers, i.e. $\mathcal{A}:\mathcal{Z}^{(\infty)}\to\mathcal{H}_{\phi}$ where $\mathcal{H}_{\phi}:=\{x\mapsto\langle\phi\left(x\right),\mathbf{w}\rangle\mid\mathbf{w}\in\mathcal{K}\}$ and $\phi:\mathcal{X}\to\mathcal{K}\subseteq\ell_2^n$ is known as the feature mapping. Let us assume that $l\left(h\left(x\right),y\right)=l_{0-1}\left(h\left(x\right),y\right):=\mathbb{I}_{yh(x)\leq0}$. Classical VC generalisation error bounds exploit the fact that $\mathrm{VCDim}\left(\mathcal{H}_{\phi}\right)=n$ and (7). In the luckiness framework of Shawe-Taylor et al. [8] it has been shown that we can use $\mathrm{fat}_{\mathcal{H}_{\phi}}\left(\gamma_z\left(\mathbf{w}\right)\right)\leq\left(\gamma_z\left(\mathbf{w}\right)\right)^{-2}$ (at the price of an extra $\log_2\left(32m\right)$ factor) in place of $\mathrm{VCDim}\left(\mathcal{H}_{\phi}\right)$ where $\gamma_z\left(\mathbf{w}\right)=\min_{(x_i,y_i)\in z}y_i\left\langle\phi\left(x_i\right),\mathbf{w}\right\rangle/\left\|\mathbf{w}\right\|$ is known as the margin. Now, the maximum margin algorithm finds the weight vector $\mathbf{w}_{\mathrm{MM}}$ that maximises $\gamma_z\left(\mathbf{w}\right)$. It is known that $\mathbf{w}_{\mathrm{MM}}$ can be written as a linear combination of the $\phi\left(x_i\right)$. For notational convenience, we shall assume that $\mathcal{A}:\mathcal{Z}^{(\infty)}\to\mathbb{R}^{(\infty)}$ maps

to the expansion coefficients $\boldsymbol{\alpha}$ such that $\|\mathbf{w}_{\boldsymbol{\alpha}}\| = 1$ where $\mathbf{w}_{\boldsymbol{\alpha}} := \sum_{i=1}^{|z|} \alpha_i \boldsymbol{\phi}(x_i)$. Our new margin bound follows from the following theorem together with (6).

---

**Theorem 2.** *Let $\epsilon_i(\boldsymbol{x})$ be the smallest $\epsilon > 0$ such that $\{\boldsymbol{\phi}(x_1), \ldots, \boldsymbol{\phi}(x_m)\}$ can be covered by at most $i$ balls of radius less than or equal $\epsilon$. Let $\Gamma_{\boldsymbol{z}}(\mathbf{w})$ be defined by $\Gamma_{\boldsymbol{z}}(\mathbf{w}) := \min_{(x_i, y_i) \in \boldsymbol{z}} \frac{y_i \langle \boldsymbol{\phi}(x_i), \mathbf{w}\rangle}{\|\boldsymbol{\phi}(x_i)\| \cdot \|\mathbf{w}\|}$. For the zero-one loss $l_{0-1}$ and the maximum margin algorithm $\mathcal{A}$, the luckiness function*

$$L(\mathcal{A}, \boldsymbol{z}) = -\min \left\{ i \in \mathbb{N} \;\middle|\; i \geq \left( \frac{\epsilon_i(\boldsymbol{x}) \sum_{j=1}^{m} |\mathcal{A}(\boldsymbol{z})_j|}{\Gamma_{\boldsymbol{z}}(\mathbf{w}_{\mathcal{A}(\boldsymbol{z})})} \right)^2 \right\}, \qquad (8)$$

*is $\omega$-small at scale $1/2m$ w.r.t. the function*

$$\omega\left(L_0, l, m, \delta, \frac{1}{2m}\right) = \left(\frac{2em}{-L_0}\right)^{-2L_0}. \qquad (9)$$

---

*Proof (Sketch).* First we note that by a slight refinement of a theorem of Makovoz [7] we know that for any $\boldsymbol{z} \in \mathcal{Z}^m$ there exists a weight vector $\tilde{\mathbf{w}} = \sum_{i=1}^{m} \tilde{\alpha}_i \boldsymbol{\phi}(x_i)$ such that

$$\left\| \tilde{\mathbf{w}} - \mathbf{w}_{\mathcal{A}(\boldsymbol{z})} \right\|^2 \leq \Gamma_{\boldsymbol{z}}^2 \left( \mathbf{w}_{\mathcal{A}(\boldsymbol{z})} \right) \qquad (10)$$

and $\tilde{\boldsymbol{\alpha}} \in \mathbb{R}^m$ has no more than $-L(\mathcal{A}, \boldsymbol{z})$ non-zero components. Although only $\mathbf{w}_{\mathcal{A}(\boldsymbol{z})}$ is of unit length, one can show that (10) implies that

$$\left\langle \mathbf{w}_{\mathcal{A}(\boldsymbol{z})}, \tilde{\mathbf{w}}/ \|\tilde{\mathbf{w}}\| \right\rangle \geq \sqrt{1 - \Gamma_{\boldsymbol{z}}^2 \left( \mathbf{w}_{\mathcal{A}(\boldsymbol{z})} \right)}.$$

Using equation (10) of [4] this implies that $\tilde{\mathbf{w}}$ correctly classifies $\boldsymbol{z} \in \mathcal{Z}^m$. Consider a fixed double sample $\boldsymbol{z} \in \mathcal{Z}^{2m}$ and let $k_0 := L(\mathcal{A}, (z_1, \ldots, z_m))$. By virtue of (3) and the aforementioned argument we only need to consider permutations $\pi_i$ such that there exists a weight vector $\tilde{\mathbf{w}} = \sum_{j=1}^{m} \tilde{\alpha}_j \boldsymbol{\phi}(x_j)$ with no more than $k_0$ non-zero $\tilde{\alpha}_j$. As there are exactly $\binom{2m}{d}$ distinct choices of $d \in \{1, \ldots, k_0\}$ training examples from the $2m$ examples $\boldsymbol{z}$ there are no more than $(2em/k_0)^{k_0}$ different subsamples to be used in $\tilde{\mathbf{w}}$. For each particular subsample $\overline{\boldsymbol{z}} \subseteq \boldsymbol{z}$ the weight vector $\tilde{\mathbf{w}}$ is a member of the class of linear classifiers in a $k_0$ (or less) dimensional space. Thus, from (7) it follows that for the given subsample $\overline{\boldsymbol{z}}$ there are no more $(2em/k_0)^{k_0}$ different dichotomies induced on the double sample $\boldsymbol{z} \in \mathcal{Z}^{2m}$. As this holds for any double sample, the theorem is proven. $\qquad \square$

There are several interesting features about this margin bound. Firstly, observe that $\sum_{j=1}^{m} |\mathcal{A}(\boldsymbol{z})_j|$ is a measure of sparsity of the solution found by the maximum margin algorithm which, in the present case, is combined with margin. Note that for normalised data, i.e. $\|\boldsymbol{\phi}(\cdot)\| = \text{constant}$, the two notion of margins coincide, i.e. $\Gamma_{\boldsymbol{z}}(\mathbf{w}) = \gamma_{\boldsymbol{z}}(\mathbf{w})$. Secondly, the quantity $\epsilon_i(\boldsymbol{x})$ can be considered as a measure of the distribution of the mapped data points in feature space. Note that for all $i \in \mathbb{N}$, $\epsilon_i(\boldsymbol{x}) \leq \epsilon_1(\boldsymbol{x}) \leq \max_{j \in \{1, \ldots, m\}} \|\boldsymbol{\phi}(x_j)\|$. Supposing that the two class-conditional probabilities $\mathbf{P}_{X|Y=y}$ are highly clustered, $\epsilon_2(\boldsymbol{x})$ will be very small. An extension of this reasoning is useful in the multi-class case; binary maximum margin classifiers are often used to solve multi-class problems [9]. There appears to be also a close relationship of $\epsilon_i(\boldsymbol{x})$ with the notion of kernel alignment recently introduced in [3]. Finally, one can use standard entropy number techniques to bound $\epsilon_i(\boldsymbol{x})$ in terms of eigenvalues of the inner product matrix or its centred variants. It is worth mentioning that although our aim was to study the maximum margin algorithm the

above theorem actually holds for any algorithm whose solution can be represented as a linear combination of the data points.

## 5 Conclusions

In this paper we have introduced a new theoretical framework to study the generalisation error of learning algorithms. In contrast to previous approaches, we considered specific learning algorithms rather than specific hypothesis spaces. We introduced the notion of algorithmic luckiness which allowed us to devise data dependent generalisation error bounds. Thus we were able to relate the compression framework of Littlestone and Warmuth with the VC framework. Furthermore, we presented a new bound for the maximum margin algorithm which not only exploits the margin but also the distribution of the *actual* training data in feature space. Perhaps the most appealing feature of our margin based bound is that it naturally combines the three factors considered important for generalisation with linear classifiers: margin, sparsity and the distribution of the data. Further research is concentrated on studying Bayesian algorithms and the relation of algorithmic luckiness to the recent findings for stable learning algorithms [2].

**Acknowledgements** This work was done while RCW was visiting Microsoft Research Cambridge. This work was also partly supported by the Australian Research Council. RH would like to thank Olivier Bousquet for stimulating discussions.

## References

[1] M. Anthony and P. Bartlett. *A Theory of Learning in Artificial Neural Networks.* Cambridge University Press, 1999.

[2] O. Bousquet and A. Elisseeff. Algorithmic stability and generalization performance. In T. K. Leen, T. G. Dietterich, and V. Tresp, editors, *Advances in Neural Information Processing Systems 13*, pages 196–202. MIT Press, 2001.

[3] N. Cristianini, A. Elisseeff, and J. Shawe-Taylor. On optimizing kernel alignment. Technical Report NC2-TR-2001-087, NeuroCOLT, http://www.neurocolt.com, 2001.

[4] R. Herbrich and T. Graepel. A PAC-Bayesian margin bound for linear classifiers: Why SVMs work. In T. K. Leen, T. G. Dietterich, and V. Tresp, editors, *Advances in Neural Information Processing Systems 13*, pages 224–230, Cambridge, MA, 2001. MIT Press.

[5] R. Herbrich and R. C. Williamson. Algorithmic luckiness. Technical report, Microsoft Research, 2002.

[6] N. Littlestone and M. Warmuth. Relating data compression and learnability. Technical report, University of California Santa Cruz, 1986.

[7] Y. Makovoz. Random approximants and neural networks. *Journal of Approximation Theory*, 85:98–109, 1996.

[8] J. Shawe-Taylor, P. L. Bartlett, R. C. Williamson, and M. Anthony. Structural risk minimization over data-dependent hierarchies. *IEEE Transactions on Information Theory*, 44(5):1926–1940, 1998.

[9] V. Vapnik. *Statistical Learning Theory*. John Wiley and Sons, New York, 1998.

[10] V. N. Vapnik and A. Y. Chervonenkis. On the uniform convergence of relative frequencies of events to their probabilities. *Theory of Probability and its Applications*, 16(2):264–281, 1971.